# Motivated Reinforcement Learning

**Peter Dayan**
Gatsby Computational Neuroscience Unit
17 Queen Square, London, England, WC1N 3AR.
dayan@gatsby.ucl.ac.uk

## Abstract

The standard reinforcement learning view of the involvement of neuromodulatory systems in instrumental conditioning includes a rather straightforward conception of motivation as prediction of sum future reward. Competition between actions is based on the motivating characteristics of their consequent states in this sense. Substantial, careful, experiments reviewed in Dickinson & Balleine,[12,13] into the neurobiology and psychology of motivation shows that this view is incomplete. In many cases, animals are faced with the choice not between many different actions at a given state, but rather whether a single response is worth executing at all. Evidence suggests that the motivational process underlying this choice has different psychological and neural properties from that underlying action choice. We describe and model these motivational systems, and consider the way they interact.

## 1 Introduction

Reinforcement learning (RL[28]) bears a tortuous relationship with historical and contemporary ideas in classical and instrumental conditioning. Although RL sheds important light in some murky areas, it has paid less attention to research concerning the *motivation* of stimulus-response (SR) links. RL methods are mainly concerned with preparatory Pavlovian (*eg* secondary) conditioning, and, in instrumental conditioning, the competition between multiple possible actions given a particular stimulus or state, based on the future rewarding or punishing consequences of those actions. These have been used to build successful and predictive models of the activity of monkey dopamine cells in conditioning.[22,24] By contrast, SR research starts from the premise that, in many circumstances, given an unconditioned stimulus (US; such as a food pellet), there is only one natural set of actions (the *habit* of approaching and eating the food), and the main issue is whether this set is worth executing (yes, if hungry, no if sated). This is traditionally conceived as a question of consummatory motivation. SR research goes on to study how these habits, and also the motivation associated with them, are 'attached' in an appropriately preparatory sense to conditioned stimuli (CSs) that are predictive of the USs.

The difference between RL's competition between multiple actions and SR's motivation of a single action might seem trivial, particularly if an extra, null, action is included in the action competition in RL, so the subject can actively choose to do nothing. However, there is substantial evidence from experi-

ments in which drive states (*eg* hunger, thirst) are manipulated, that motivation in the SR sense works in a sophisticated, intrinsically goal-sensitive, way and can exert unexpected effects on instrumental conditioning. By comparison with RL, psychological study of multiple goals within single environments is quite advanced, particularly in experiments in which one goal or set of goals is effective during learning, and another during performance. Based on these and other studies, (and earlier theoretical ideas from, amongst others, Konorski,[18,19] Dickinson, Balleine and their colleagues[13] have suggested that there are really two separate motivational systems, one associated with Pavlovian motivation, as in SR, and one associated with instrumental action choice. They further suggest, partly based on related suggestions by Berridge and his colleagues,[7] that only the Pavlovian system involves dopamine. Neither the Pavlovian nor the instrumental system maps cleanly onto the standard view of RL, and the suggestion about dopamine would clearly significantly damage the RL interpretation of the involvement of this neuromodulatory system in conditioning.

In this paper, we describe some of the key evidence supporting the difference between instrumental and Pavlovian motivation (see also Balkenius[3] and Spier[25]), and expand the model of RL in the brain to incorporate SR motivation and concomitant evidence on intrinsic goal sensitivity (as well as intrinsic habits). Some of the *computational* properties of this new model turn out to be rather strange – but this is a direct consequence of equivalently strange observable behavior.

## 2  Theoretical and Experimental Background

Figure 1 shows a standard view of the involvement of the dopamine system in RL.[22,24] Dopamine neurons in the ventral tegmental area (VTA) and substantia nigra pars compacta (SNc) report the temporal difference (TD) error $\delta(t)$. In the simplest version of the theory, this is calculated as $\delta(t) = r(t) + V^\pi(\mathbf{x}(t+1)) - V^\pi(\mathbf{x}(t))$, where $r(t)$ is the value of the reward at time $t$, $\mathbf{x}(t)$ is an internal representation of the state at time $t$, $V^\pi(\mathbf{x}(t))$ is the expectation of the sum total future reward expected by the animal based on starting from that state, following policy $\pi$, and the transition from $\mathbf{x}(t)$ to $\mathbf{x}(t+1)$ is occasioned by the action $a$ selected by the subject. In the actor-critic[6] version of the dopamine theory, this TD error signal is put to two uses. One is adapting parameters that underlie the actual predictions $V^\pi(\mathbf{x}(t))$. For this, $\delta(t) > 0$ if the prediction from the state at time $t$, $V^\pi(\mathbf{x}(t))$, is overly pessimistic with respect to the sum of the actual reward, $r(t)$, and the estimated future reward, $V^\pi(\mathbf{x}(t+1))$, from the subsequent state. The other use for $\delta(t)$ is criticizing the action $a$ adopted at time $t$. For this, $\delta(t) > 0$ implies that the action chosen is worth more than the average worth of $\mathbf{x}(t)$, and that the overall policy $\pi$ of the subject can therefore be improved by choosing it more often. In a Q-learning[31] version of the theory, $Q^\pi(\mathbf{x}, a)$ values are learned using an analogous quantity to $\delta(t)$, for each pair of states $\mathbf{x}$ and actions $a$, and can directly be used to choose between the actions to improve the policy.

Even absent an account for intrinsic habits, three key paradigms show the incompleteness of this view of conditioning: appetitive Pavlovian-instrumental transfer,[15] intrinsic drive preference under specific deprivation states,[8] and incentive learning, as in the control of chains of instrumental behavior.[5]

The SR view of conditioning places its emphasis on motivational control of a prepotent action. That is, the natural response associated with a stimulus (presumably as output by an action specification mechanism) is only elicited if

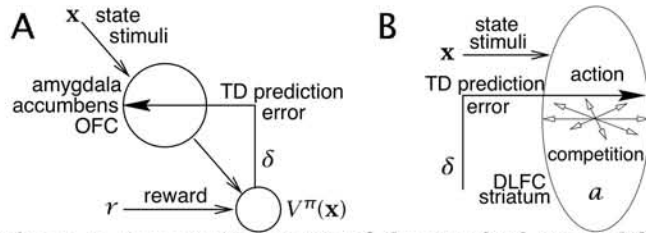

**Figure 1:** Actor-critic version of the standard RL model. A) **Evaluator:** A TD error signal $\delta$ to learn $V^{\pi}(\mathbf{x})$ to match the sum of future rewards $r$, putatively via the basolateral nuclei of the amygdala, the orbitofrontal cortex and the nucleus accumbens. B) **Instrumental controller:** The TD error $\delta$ is used to choose, and teach the choice of, appropriate actions $a$ to maximize future reward based on the current state and stimuli, putatively via the dorsolateral prefrontal cortex and the dorsal striatum.

it is motivationally appropriate, according to the current goals of the animal. The suggestion is that this is mediated by a separate motivational system. USs have direct access to this system, and CSs have learned access. A conclusion used to test this structure for the control of actions is that this motivational system could be able to energize *any* action being executed by the animal. Appetitive Pavlovian-instrumental transfer[15] shows exactly this. Animals executing an action for an outcome under instrumental control, will perform more quickly when a CS predictive of reward is presented, even if the CS predicts a completely different reward from the instrumental outcome. This effect is abolished by lesions of the shell of the nucleus accumbens,[10] one of the main targets of DA from the VTA. The standard RL model offers no account of the speed or force of action (though one could certainly imagine various possible extensions), and has no natural way to accommodate this finding.*

The second challenge to RL comes from experiments on the effects of changing specific and general needs for animals. For instance, Berridge & Schulkin[8] first gave rats sucrose and saline solutions with one of a bitter (quinine) and a sour (citric) taste. They then artificially induced a strong physiological requirement for salt, for the first time in the life of the animal. Presented with a choice between the two flavors (in plain water, *ie* in extinction), the rats preferred to drink the flavor associated with the salt. Furthermore, the flavor paired with the salt was awarded positive hedonic reactions, whereas before pairing (and if it had been paired with sucrose instead) it was treated as being aversive. The key feature of this experiment is that this preference is evident without the opportunity for learning. Whereas the RL system could certainly take the physiological lack of salt as helping determine part of the state $\mathbf{x}(t)$, this could only exert an effect on behavior through learning, contrary to the evidence.

The final complexity for standard RL comes from incentive learning. One paradigm involves a sequential chain of two actions ($a_1$ and $a_2$) that rats had to execute in order to get a reward.[5] The subjects were made hungry, and were first trained to perform action $a_2$ to get a particular reward (a Noyes pellet), and then to perform the chain of actions $a_1 \rightarrow a_2$ to get the reward. In a final test phase, the animals were offered the chance of executing $a_1$ and $a_2$ in extinction, for half of them when they were still hungry; for the other half when they were sated on their normal diet. Figure 2A shows what happens. Sated animals perform $a_1$ at the same rate as hungry animals, but perform $a_2$ sig-

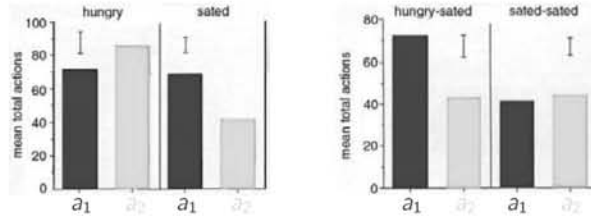

**Figure 2**: Incentive learning. A) Mean total actions $a_1$ and $a_2$ for an animal trained on the chain schedule $a_1 \to a_2 \to$ Noyes pellets. Hungry and sated rats perform $a_1$ at the same rate, but sated animals fail to perform $a_2$. B) Mean total actions when sated following prior re-exposure to the Noyes pellets when hungry ('hungry-sated') or when sated ('sated-sated'). Animals re-exposed when sated are significantly less willing to perform $a_2$. Note the change in scale between A and B. Adapted from Balleine *et al.*[5]

nificantly less frequently. Figure 2B shows the basic incentive learning effect. Here, before the test, animals were given a limited number of the Noyes pellets (without the availability of the manipulanda) either when hungry or when sated. Those who experienced them hungry ('hungry-sated') show the same results as the 'sated' group of figure 2A; whereas those who experienced them sated ('sated-sated') now declined to perform action $a_1$ either.

This experiment makes two points about the standard RL model. First, the action nearest to the reward ($a_2$) is affected by the deprivation state without additional learning. This is like the effect of specific deprivation states discussed above. Second is that a change in the willingness to execute $a_1$ happens after re-exposure to the Noyes pellets whilst sated; this learning is believed to involve insular cortex (part of gustatory neocortex[4]). That re-exposure directly affects the choice of $a_1$ suggests that the instrumental act is partly determined by an evaluation of its ultimate consequence, a conclusion that relates to a long-standing psychological debate about the 'cognitive' evaluation of actions. Dickinson & Balleine[13] suggest that the execution of $a_2$ is mainly controlled by Pavlovian contingencies, and that Pavlovian motivation is instantly sensitive to goal devaluation via satiation. At this stage in the experiment, however, $a_1$ is controlled by instrumental contingencies. By comparison with Pavlovian motivation, instrumental motivation is powerful (since it can depend on response-outcome expectancies), but dumb (since, without re-exposure, the animal works hard doing $a_1$ when it wouldn't be interested in the food in any case). Ultimately, after extended training,[14] in the birth of a new habit, $a_1$ becomes controlled by Pavlovian contingencies too, and so becomes directly sensitive to devaluation.[†]

## 3  New Model

These experiments suggest some major modifications to the standard RL view. Figure 3 shows a sketch of the new model, whose key principles include

- Pavlovian motivation (figure 3A) is associated with prediction error
$$\delta(t) = r(t) + V^\pi(\mathbf{x}(t+1)) - V^\pi(\mathbf{x}(t)$$
for long term expected future rewards $V^\pi(\mathbf{x}, a)$, given a policy $\pi$. Adopting this makes the model account for the classical conditioning paradigms explained by the standard RL model.

---

[†]It is not empirically clear whether actions that have become habits are completely automatic[1] or are subject to Pavlovian motivational influences.

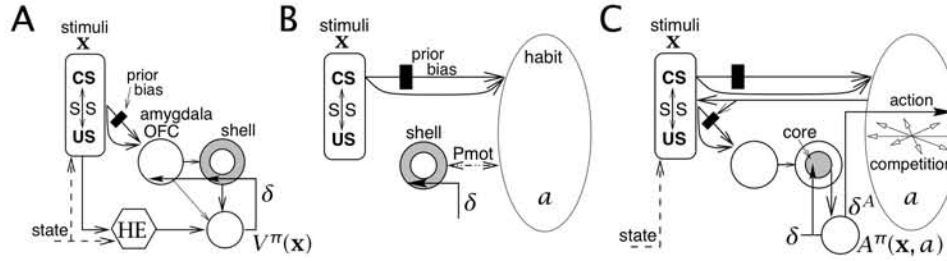

**Figure 3**: Tripartite model. A) **Evaluator:** USs are evaluated by a hard-wired evaluation system (HE) which is intrinsically sensitive to devaluation. USs can also be evaluated via a plastic route, as in figure 1, but which nevertheless has prior biases. CSs undergo Pavlovian stimulus substitution with the USs they predict, and can also be directly evaluated through the learned route. The two sources of information for $V^\pi(\mathbf{x})$ *compete*, forcing the plastic route to adjust to the hard-wired route. B) **Habit system:** The SR mapping suggests an appropriate action based on the state $\mathbf{x}$; the vigor of its execution is controlled by dopaminergic $\delta$, putatively acting via the shell of the accumbens. C) **Instrumental controller:** Action choice is based on advantages, which are learned, putatively via the core of the accumbens. Prefrontal working memory is used to unfold the consequences of chosen actions.

- $r(t)$ is determined by a devaluation-sensitive, 'hard-wired', US evaluator that provides direct value information about motivationally inappropriate USs.
- $\delta(t)$, possibly acting through the shell of the accumbens, provides Pavlovian motivation for pre-wired and new habits (figure 3B), as in Pavlovian instrumental transfer.
- $V^\pi(\mathbf{x}(t))$ is determined by two competing sources: one as in the standard model (involving the basolateral nuclei of the amygdala and the orbitofrontal cortex (oFC),[16,23] and including prior biases (sweet tasting foods are appetitive) expressed in the connections from primary taste cortex to oFC and the amygdala; the other, which is primary, dependent largely on a stimulus substitution[20] relationship between CSs and USs, that is also devaluation-dependent. The latter is important for ultimate Pavlovian control over actions; the former for phenomena such as secondary conditioning, which are known to be devaluation independent.[17] Figure 4A (dashed) shows the contribution of the hard-wired evaluation route, via stimulus-substitution, on the prediction of value in classical conditioning. Here, stimulus-substitution was based on a form of Hebbian learning with a synaptic trace, so the shorter the CS-US interval, the greater the HE component. This translates into greater immediate sensitivity to devaluation, the main characteristic of the hard-wired route. The plastic route via the amygdala takes responsibility for the remainder of the prediction; and the sum prediction is always correct (solid line).
- Short-term storage of predictive stimuli in prefrontal working memory is gated[9] by $\delta(t)$, so can also be devaluation dependent.
- Instrumental motivation depends on policy-based advantages (3C; Baird[2])
$$A^\pi(\mathbf{x}, a) = Q^\pi(\mathbf{x}, a) - V^\pi(\mathbf{x})$$
trained by the error signal
$$\delta^A(t) = \delta(t) - A^\pi(\mathbf{x}, a)$$
Over the course of policy improvement, the advantage of a sub-optimal action becomes negative, and of an optimal action tends to 0. The latter offers a natural model of the transition from instrumental action selection to an SR habit. Note that, in this actor-critic scheme, some aspects of advantages are not necessary, such as the normalizing updates.

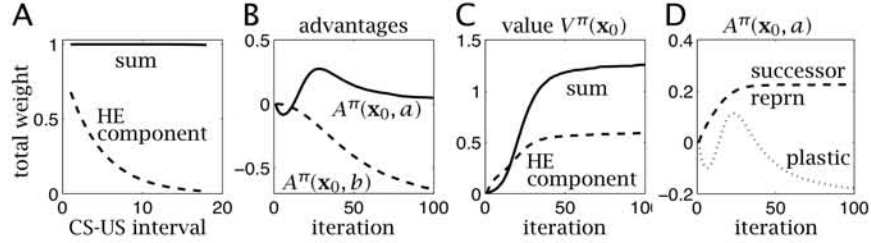

**Figure 4**: A) Role of the hard-wired route (dashed line), via stimulus-substitution, in predicting future reward ($r=1$) as a function of the CS-US interval. The solid line shows that the net prediction is always correct. B) Advantages of useful ($a$) and worthless ($b$) actions at the start state $\mathbf{x}_0$. C) Evolution of the value of $\mathbf{x}_0$ over learning. The solid line shows the mean value; the dashed line the hard-wired component, providing immediate devaluation sensitivity. D) Construction of $A^\pi(\mathbf{x}_0, a)$ via a successor representation component[11] (dashed) and a conventionally learned component (dotted). The former is sensitive to re-exposure devaluation, as in figure 2B. B-D) Action $a$ produces reward $r=1$ with probability 0.9 after 3 timesteps; curves are averages over 2000 runs.

Figure 4B;C show two aspects of instrumental conditioning. Two actions compete at state $\mathbf{x}_0$, one, $a$, with a small cost and a large future payoff; the other, $b$, with no cost and no payoff. Figure 4B shows the development of the advantages of these actions over learning. Action $a$ starts looking worse, because it has a greater immediate cost; its advantage increases as the worth of $a$ grows greater than the mean value of $\mathbf{x}_0$, and then goes to 0 (the birth of the habit) as the subject learns to choose it every time. Figure 4C shows the value component of state $\mathbf{x}_0$. This comes to be responsible for the entire prediction (as $A^\pi(\mathbf{x}_0, a) \to 0$). As in figure 4A, there is a hard-wired component to this value which would result in the immediate decrement of response evident in figure 2A.

- On-line action choice is dependent on $\delta^A(t)$ as in learned klinokinesis.[21] Incentive learning in chains suggests that the representation underlying the advantage of an action includes information about its future consequences, either through an explicit model,[27,29] a successor representation,[11] or perhaps a form of $\beta$-model.[26] One way of arranging this would use a VTE-like[30] mechanism for proposing actions (perhaps using working memory in prefrontal cortex), in order to test their advantages. Figure 4D shows the consequence of using a learned successor representation underlying the advantage $A^\pi(\mathbf{x}_0, a)$ shown in figure 4B. The dashed line shows the component of $A^\pi(\mathbf{x}_0, a)$ dependent on a learned successor representation, and the prior bias about the value of the reward, and which is therefore sensitive to re-exposure (when the value accorded to the reward is decreased); the dotted line shows the remaining component of $A^\pi(\mathbf{x}_0, a)$, learned in the standard way. Re-exposure sensitivity (*ie* incentive learning) will exist over roughly iterations $25 - 75$.

- SR models also force consideration of the repertoire of possible actions or responses available at a given state (figure 3B;C). We assume that both cortico-cortical and cortico-(dorsal) striatal plasticity sculpt this collection, using $\delta^A(t)$ directly, and maybe also correlational learning rules.

The details of the model are not experimentally fully determined, although its general scheme is based quite straightforwardly from the experimental evidence referred to (and many other experiments), and by consistency with the activity of dopamine cells (recordings of which have so far used only a single motivational state).

## 4  Discussion

Experiments pose a critical challenge to our understanding of the psychological and neural implementation of reinforcement learning,[12,13] suggesting the importance of two different sorts of motivation in controlling behavior. With both empirical and theoretical bases, we have put these two aspects together through the medium of advantages. The most critical addition is a hard-wired, stimulus-substitution sensitive, route for the evaluation of stimuli and states, which competes with a plastic route through the amygdala and the oFC. This hard-wired route has the property of intrinsic sensitivity to various sorts of devaluation, and this leads to motivationally appropriate behavior. The computational basis of the new aspects of the model focus on motivational control of SR links (via $V^\pi$), to add to motivational control of instrumental actions (via $A^\pi$). We also showed the potential decomposition of the advantages into a component based on the successor representation and therefore sensitive to re-exposure as in incentive learning, and a standard, learned, component.

The model is obviously incomplete, and requires testing in richer environments. In particular, we have yet to explore how habits get created from actions as the maximal advantage goes to 0.

### Acknowledgements

I am very grateful to Christian Balkenius, Bernard Balleine, Tony Dickinson, Sham Kakade, Emmet Spier and Angela Yu for discussions. Funding was from the Gatsby Charitable Foundation.

## Footnotes

*Note that aversive Pavlovian instrumental transfer, in the form of the suppression of appetitive instrumental responding, is the conventional method for testing aversive Pavlovian conditioning. There is an obvious motivational explanation for this as well as the conventional view of competition between appetitive and protective actions.

## References

[1] Adams, CD (1982) Variations in the sensitivity of instrumental responding to reinforcer devaluation. *QJEP* **34B**:77-98.

[2] Baird, LC (1993) *Advantage Updating.* Technical report WL-TR-93-1146, Wright-Patterson Air Force Base.

[3] Balkenius, C (1995) *Natural Intelligence in Artificial Creatures.* PhD Thesis, Department of Cognitive Science, Lund University, Sweden.

[4] Balleine, BW & Dickinson, A (1998) Goal-directed instrumental action: Contingency and incentive learning and their cortical substrates. *Neuropharmacology* **37**:407-419.

[5] Balleine, BW, Garner, C, Gonzalez, F & Dickinson, A (1995) Motivational control of heterogeneous instrumental chains. *Journal of Experimental Psychology: Animal Behavior Processes* **21**:203-217.

[6] Barto, AG, Sutton, RS & Anderson, CW (1983) Neuronlike elements that can solve difficult learning problems. *IEEE SMC* **13**:834-846.

[7] Berridge, KC (2000) Reward learning: Reinforcement, incentives, and expectations. In DL Medin, editor, *The Psychology of Learning and Motivation* **40**:223-278.

[8] Berridge, KC & Schulkin, J (1989) Palatability shift of a salt-associated incentive during sodium depletion. *Quarterly Journal of Experimental Psychology: Comparative & Physiological Psychology* **41**:121-138.

[9] Braver, TS, Barch, DM & Cohen, JD (1999) Cognition and control in schizophrenia: A computational model of dopamine and prefrontal function. *Biological Psychiatry* **46**:312-328.

[10] Corbit, LH, Muir, JL & Balleine, BW (2001) The role of the nucleus accumbens in instrumental conditioning: Evidence of a functional dissociation between accumbens core and shell. *Journal of Neuroscience* **21**:3251-3260.

[11] Dayan, P (1993) Improving generalisation for temporal difference learning: The successor representation. *Neural Computation* **5**:613-624.

[12] Dickinson, A & Balleine, B (1994) Motivational control of goal-directed action. *Animal Learning & Behavior* **22**:1-18.

[13] Dickinson, A & Balleine, B (2001) The role of learning in motivation. In CR Gallistel, editor, *Learning, Motivation and Emotion, Volume 3 of Steven's Handbook of Experimental Psychology, Third Edition*. New York, NY: Wiley.

[14] Dickinson, A, Balleine, B, Watt, A, Gonzalez, F & Boakes, RA (1995) Motivational control after extended instrumental training. *Animal Learning & Behavior* **23**:197-206.

[15] Estes, WK (1943). Discriminative conditioning. I. A discriminative property of conditioned anticipation. *JEP* **32**:150-155.

[16] Holland, PC & Gallagher, M (1999) Amygdala circuitry in attentional and representational processes. *Trends in Cognitive Sciences* **3**:65-73.

[17] Holland, PC & Rescorla, RA (1975) The effect of two ways of devaluing the unconditioned stimulus after first- and second-order appetitive conditioning. *Journal of Experimental Psychology: Animal Behavior Processes* **1**:355-363.

[18] Konorski, J (1948) *Conditioned Reflexes and Neuron Organization*. Cambridge, England: Cambridge University Press.

[19] Konorski, J (1967) *Integrative Activity of the Brain: An Interdisciplinary Approach*. Chicago, IL: University of Chicago Press.

[20] Mackintosh, NJ (1974) *The Psychology of Animal Learning*. New York, NY: Academic Press.

[21] Montague, PR, Dayan, P, Person, C & Sejnowski TJ (1995) Bee foraging in uncertain environments using predictive hebbian learning. *Nature* **377**:725-728.

[22] Montague, PR, Dayan, P & Sejnowski, TJ (1996) A framework for mesencephalic dopamine systems based on predictive Hebbian learning. *Journal of Neuroscience* **16**:1936-1947.

[23] Schoenbaum, G, Chiba, AA & Gallagher, M (1999) Neural encoding in orbitofrontal cortex and basolateral amygdala during olfactory discrimination learning. *Journal of Neuroscience* **19**:1876-1884.

[24] Schultz, W, Dayan, P & Montague, PR (1997) A neural substrate of prediction and reward. *Science* **275**:1593-1599.

[25] Spier, E (1997) *From Reactive Behaviour to Adaptive Behaviour*. PhD Thesis, Balliol College, Oxford.

[26] Sutton, RS (1995) TD models: modeling the world at a mixture of time scales. In A Prieditis & S Russell, editors, *Proceedings of the Twelfth International Conference on Machine Learning*. San Francisco, CA: Morgan Kaufmann, 531-539.

[27] Sutton, RS & Barto, AG (1981) An adaptive network that constructs and uses an internal model of its world. *Cognition and Brain Theory* **4**:217-246.

[28] Sutton, RS & Barto, AG (1998) *Reinforcement Learning*. Cambridge, MA: MIT Press.

[29] Sutton, RS & Pinette, B (1985) The learning of world models by connectionist networks. *Proceedings of the Seventh Annual Conference of the Cognitive Science Society*. Irvine, CA: Lawrence Erlbaum, 54-64.

[30] Tolman, EC (1938) The determiners of behavior at a choice point. *Psychological Review* **45**:1-41.

[31] Watkins, CJCH (1989) *Learning from Delayed Rewards*. PhD Thesis, University of Cambridge, Cambridge, UK.
